# Statistical Analysis of Semi-Supervised Regression

**John Lafferty**
Computer Science Department
Carnegie Mellon University
Pittsburgh, PA 15213
lafferty@cs.cmu.edu

**Larry Wasserman**
Department of Statistics
Carnegie Mellon University
Pittsburgh, PA 15213
larry@stat.cmu.edu

## Abstract

Semi-supervised methods use unlabeled data in addition to labeled data to construct predictors. While existing semi-supervised methods have shown some promising empirical performance, their development has been based largely based on heuristics. In this paper we study semi-supervised learning from the viewpoint of minimax theory. Our first result shows that some common methods based on regularization using graph Laplacians do not lead to faster minimax rates of convergence. Thus, the estimators that use the unlabeled data do not have smaller risk than the estimators that use only labeled data. We then develop several new approaches that provably lead to improved performance. The statistical tools of minimax analysis are thus used to offer some new perspective on the problem of semi-supervised learning.

## 1 Introduction

Suppose that we have labeled data $\mathcal{L} = \{(X_1, Y_1), \ldots (X_n, Y_n)\}$ and unlabeled data $\mathcal{U} = \{X_{n+1}, \ldots X_N\}$ where $N \gg n$ and $X_i \in \mathbb{R}^D$. Ordinary regression and classification techniques use $\mathcal{L}$ to predict $Y$ from $X$. Semi-supervised methods also use the unlabeled data $\mathcal{U}$ in an attempt to improve the predictions. To justify these procedures, it is common to invoke one or both of the following assumptions:

> **Manifold Assumption (M):** The distribution of $X$ lives on a low dimensional manifold.

> **Semi-Supervised Smoothness Assumption (SSS):** The regression function $m(x) = \mathbb{E}Y \mid X = x$ is very smooth where the density $p(x)$ of $X$ is large. In particular, if there is a path connecting $X_i$ and $X_j$ on which $p(x)$ is large, then $Y_i$ and $Y_j$ should be similar with high probability.

While these assumptions are somewhat intuitive, and synthetic examples can easily be constructed to demonstrate good performance of various techniques, there has been very little theoretical analysis of semi-supervised learning that rigorously shows how the assumptions lead to improved performance of the estimators.

In this paper we provide a statistical analysis of semi-supervised methods for regression, and propose some new techniques that provably lead to better inferences, under appropriate assumptions. In particular, we explore precise formulations of SSS, which is motivated by the intuition that high density level sets correspond to clusters of similar objects, but as stated above is quite vague. To the best of our knowledge, no papers have made the assumption precise and then explored its consequences in terms of rates of convergence, with the exception of one of the first papers on semi-supervised learning, by Castelli and Cover (1996), which evaluated a simple mixture model, and the recent paper of Rigollet (2006) in the context of classification. This situation is striking, given the level of activity in this area within the machine learning community; for example, the recent survey of semi-supervised learning by Zhu (2006) contains 163 references.

Among our findings are:

1. Under the manifold assumption M, the semi-supervised smoothness assumption SSS is superfluous. This point was made heuristically by Bickel and Li (2006), but we show that in fact ordinary regression methods are automatically adaptive if the distribution of $X$ concentrates on a manifold.

2. Without the manifold assumption M, the semi-supervised smoothness assumption SSS as usually defined is too weak, and current methods don't lead to improved inferences. In particular, methods that use regularization based on graph Laplacians do not achieve faster rates of convergence.

3. Assuming specific conditions that relate $m$ and $p$, we develop new semi-supervised methods that lead to improved estimation. In particular, we propose estimators that reduce bias by estimating the Hessian of the regression function, improve the choice of bandwidths using unlabeled data, and estimate the regression function on level sets.

The focus of the paper is on a theoretical analysis of semi-supervised regression techniques, rather than the development of practical new algorithms and techniques. While we emphasize regression, most of our results have analogues for classification. Our intent is to bring the statistical perspective of minimax analysis to bear on the problem, in order to study the interplay between the labeled sample size and the unlabeled sample size, and between the regression function and the data density. By studying simplified versions of the problem, our analysis suggests how precise formulations of assumptions M and SSS can be made and exploited to lead to improved estimators.

## 2 Preliminaries

The data are $(X_1, Y_1, R_1), \ldots, (X_N, Y_N, R_N)$ where $R_i \in \{0, 1\}$ and we observe $Y_i$ only if $R_i = 1$. The labeled data are $\mathcal{L} = \{(X_i, Y_i) \quad R_i = 1\}$ and the unlabeled data are $\mathcal{U} = \{(X_i, Y_i) \quad R_i = 0\}$. For convenience, assume that data are labeled so that $R_i = 1$ for $i = 1, \ldots, n$ and $R_i = 0$ for $i = n + 1, \ldots, N$. Thus, the labeled sample size is $n$, and the unlabeled sample size is $u = N - n$.

Let $p(x)$ be the density of $X$ and let $m(x) = \mathbb{E}(Y \mid X = x)$ denote the regression function. Assume that $R \perp\!\!\!\perp Y \mid X$ (missing at random) and that $R_i \mid X_i \sim \text{Bernoulli}(\pi(X_i))$. Finally, let $\mu = \mathbb{P}(R_i = 1) = \int \pi(x)p(x)dx$. For simplicity we assume that $\pi(x) = \mu$ for all $x$. The missing at random assumption $R \perp\!\!\!\perp Y \mid X$ is crucial, although this point is rarely emphasized in the machine learning literature.

It is clear that without some further conditions, the unlabeled data are useless. The key assumption we need is that there is some correspondence between the shape of the regression function $m$ and the shape of the data density $p$.

We will use minimax theory to judge the quality of an estimator. Let $\mathcal{R}$ denote a class of regression functions and let $\mathcal{F}$ denote a class of density functions. In the classical setting, we observe labeled data $(X_1, Y_2), \ldots, (X_n, Y_n)$. The pointwise minimax risk, or mean squared error (MSE), is defined by

$$R_n(x) = \inf_{\widehat{m}_n} \sup_{m \in \mathcal{R}, p \in \mathcal{F}} \mathbb{E}(\widehat{m}_n(x) - m(x))^2 \tag{1}$$

where the infimum is over all estimators. The global minimax risk is defined by

$$R_n = \inf_{\widehat{m}_n} \sup_{m \in \mathcal{R}, p \in \mathcal{F}} \mathbb{E} \int (\widehat{m}_n(x) - m(x))^2 dx. \tag{2}$$

A typical assumption is that $\mathcal{R}$ is the Sobolev space of order two, meaning essentially that $m$ has smooth second derivatives. In this case we have[1] $R_n \asymp n^{-4/(4+D)}$. The minimax rate is achieved by kernel estimators and local polynomial estimators. In particular, for kernel estimators if we use a product kernel with common bandwidth $h_n$ for each variable, choosing $h_n \sim n^{-1/(4+D)}$ yields an

estimator with the minimax rate. The difficulty is that the rate $R_n \asymp n^{-4/(4+D)}$ is extremely slow when $D$ is large.

In more detail, let $C > 0$ and let $B$ be a positive definite matrix, and define

$$\mathcal{R} = \left\{ m \quad \left| m(x) - m(x_0) - (x - x_0)^T \nabla m(x_0) \right| \leq \frac{C}{2}(x - x_0)^T B(x - x_0) \right\} \tag{3}$$

$$\mathcal{F} = \left\{ p \quad p(x) \geq b > 0, \ |p(x_1) - p(x_2)| \leq c\|x_1 - x_2\|_2^\alpha \right\}. \tag{4}$$

Fan (1993) shows that the local linear estimator is asymptotically minimax for this class. This estimator is given by $\widehat{m}_n(x) = a_0$ where $(a_0, a_1)$ minimizes $\sum_{i=1}^n (Y_i - a_0 - a_1^T(X_i - x))^2 K(H^{-1/2}(X_i - x))$, where $K$ is a symmetric kernel and $H$ is a matrix of bandwidths.

The asymptotic MSE of the local linear estimator $\widehat{m}(x)$ using the labeled data is

$$R(H) = \left( \frac{1}{2}\mu_2^2(K)\mathrm{tr}(\mathcal{H}_m(x)H) \right)^2 + \frac{1}{n|H|^{1/2}}\frac{v_0\sigma^2}{p(x)} + o(\mathrm{tr}(H)) \tag{5}$$

where $\mathcal{H}_m(x)$ is the Hessian of $m$ at $x$, $\mu_2(K) = \int K^2(u)\,du$ and $v_0$ is a constant. The optimal bandwidth matrix $H_*$ is given by

$$H_* = \left( \frac{v_0\sigma^2|\mathcal{H}_m|^{1/2}}{\mu_2^2(K)nDp(x)} \right)^{2/(D+4)} (\mathcal{H}_m)^{-1} \tag{6}$$

and $R(H_*) = O(n^{-4/(4+D)})$. This result is important to what follows, because it suggests that if the Hessian $\mathcal{H}_m$ of the regression function is related to the Hessian $\mathcal{H}_p$ of the data density, one may be able to estimate the optimal bandwidth matrix from unlabeled data in order to reduce the risk.

## 3  The Manifold Assumption

It is common in the literature to invoke both M and SSS. But if M holds, SSS is not needed. This is argued by Bickel and Li (2006) who say, "We can unwittingly take advantage of low dimensional structure without knowing it."

Suppose $X \in \mathbb{R}^D$ has support on a manifold $\mathcal{M}$ with dimension $d < D$. Let $\widehat{m}_h$ be the local linear estimator with diagonal bandwidth matrix $H = h^2 I$. Then Bickel and Li show that the bias and variance are

$$b(x) = h^2 J_1(x)(1 + o_P(1)) \quad \text{and} \quad v(x) = \frac{J_2(x)}{nh^d}(1 + o_P(1)) \tag{7}$$

for some functions $J_1$ and $J_2$. Choosing $h \asymp n^{-1/(4+d)}$ yields a risk of order $n^{-4/(4+d)}$, which is the optimal rate for data that to lie on a manifold of dimension $d$.

To use the above result we would need to know $d$. Bickel and Li argue heuristically that the following procedure will lead to a reasonable bandwidth. First, estimate $d$ using the procedure in Levina and Bickel (2005). Now let $\mathcal{B} = \{\lambda_1/n^{1/(\widehat{d}+4)}, \ldots, \lambda_B/n^{1/(\widehat{d}+4)}\}$ be a set of bandwidths, scaling the asymptotic order $n^{-1/(\widehat{d}+4)}$ by different constants. Finally, choose the bandwidth $h \in \mathcal{B}$ that minimizes a local cross-validation score.

We now show that, in fact, one can skip the step of estimating $d$. Let $E_1, \ldots, E_n$ be independent Bernoulli ($\theta = \frac{1}{2}$) random variables. Split the data into two groups, so that $I_0 = \{i \quad E_i = 0\}$ and $I_1 = \{i \quad E_i = 1\}$. Let $\mathcal{H} = \{n^{-1/(4+d)} \quad 1 \leq d \leq D\}$. Construct $\widehat{m}_h$ for $h \in \mathcal{H}$ using the data in $I_0$, and estimate the risk from $I_1$ by setting $\widehat{R}(h) = |I_1|^{-1} \sum_{i \in I_1} (Y_i - \widehat{m}_h(X_i))^2$. Finally, let $\widehat{h}$ minimize $\widehat{R}(h)$ and set $\widehat{m} = \widehat{m}_{\widehat{h}}$. For simplicity, let us assume that both $Y_i$ and $X_i$ are bounded by a finite constant $B$.

**Theorem 1.** *Suppose that and $|Y_i| \leq B$ and $|X_{ij}| \leq B$ for all $i$ and $j$. Assume the conditions in Bickel and Li (2006). Suppose that the data density $p(x)$ is supported on a manifold of dimension $d \geq 4$. Then we have that*

$$\mathbb{E}(\widehat{m}(x) - m(x))^2 = \widetilde{O}\left( \frac{1}{n^{4/(4+d)}} \right). \tag{8}$$

The notation $\widetilde{O}$ allows for logarithmic factors in $n$.

*Proof.* The risk is, up to a constant, $R(h) = \mathbb{E}(Y - \widehat{m}(X))^2$, where $(X, Y)$ is a new pair and $Y = m(X) + \epsilon$. Note that $(Y - \widehat{m}_h(X))^2 = Y^2 - 2Y\widehat{m}_h(X) + \widehat{m}_h^2(X)$, so $R(h) = \mathbb{E}(Y^2) - 2\mathbb{E}(Y\widehat{m}_h(X)) + \widehat{m}_h^2(X)$. Let $n_1 = |I_1|$. Then,

$$\widehat{R}(h) = \frac{1}{n_1}\sum_{i \in I_1} Y_i^2 - \frac{2}{n_1}\sum_{i \in I_1} Y_i\widehat{m}_h(X_i) + \frac{1}{n_1}\sum_{i \in I_1} \widehat{m}_h^2(X_i). \tag{9}$$

By conditioning on the data in $I_0$ and applying Bernstein's inequality, we have

$$\mathbb{P}\left(\max_{h \in \mathcal{H}}|\widehat{R}(h) - R(h)| > \epsilon\right) \le \sum_{h \in \mathcal{H}}\mathbb{P}\left(|\widehat{R}(h) - R(h)| > \epsilon\right) \le De^{-nc\epsilon^2} \tag{10}$$

for some $c > 0$. Setting $\epsilon_n = \sqrt{C\log n/n}$ for suitably large $C$, we conclude that

$$\mathbb{P}\left(\max_{h \in \mathcal{H}}|\widehat{R}(h) - R(h)| > \sqrt{\frac{C\log n}{n}}\right) \longrightarrow 0. \tag{11}$$

Let $h_*$ minimize $R(h)$ over $\mathcal{H}$. Then, except on a set of probability tending to 0,

$$R(\widehat{h}) \le \widehat{R}(\widehat{h}) + \sqrt{\frac{C\log n}{n}} \le \widehat{R}(h_*) + \sqrt{\frac{C\log n}{n}} \tag{12}$$

$$\le R(h_*) + 2\sqrt{\frac{C\log n}{n}} = O\left(\frac{1}{n^{4/(4+d)}}\right) + 2\sqrt{\frac{C\log n}{n}} = \widetilde{O}\left(\frac{1}{n^{4/(4+d)}}\right) \tag{13}$$

where we used the assumption $d \ge 4$ in the last equality. If $d = 4$ then $O(\sqrt{\log n/n}) = \widetilde{O}(n^{-4/(4+d)})$; if $d > 4$ then $O(\sqrt{\log n/n}) = o\left(n^{4/(4+d)}\right)$. $\square$

We conclude that ordinary regression methods are automatically adaptive, and achieve the low-dimensional minimax rate if the distribution of $X$ concentrates on a manifold; there is no need for semi-supervised methods in this case. Similar results apply to classification.

## 4 Kernel Regression with Laplacian Regularization

In practice, it is unlikely that the distribution of $X$ would be supported exactly on a low-dimensional manifold. Nevertheless, the shape of the data density $p(x)$ might provide information about the regression function $m(x)$, in which case the unlabeled data are informative.

Several recent methods for semi-supervised learning attempt to exploit the smoothness assumption SSS using regularization operators defined with respect to graph Laplacians (Zhu et al., 2003; Zhou et al., 2004; Belkin et al., 2005). The technique of Zhu et al. (2003) is based on Gaussian random fields and harmonic functions defined with respect to discrete Laplace operators. To express this method in statistical terms, recall that standard kernel regression corresponds to the locally constant estimator

$$\widehat{m}_n(x) = \arg\min_{m(x)} \sum_{i=1}^n K_h(X_i, x)(Y_i - m(x))^2 = \frac{\sum_{i=1}^n K_h(X_i, x)Y_i}{\sum_{i=1}^n K_h(X_i, x)} \tag{14}$$

where $K_h$ is a symmetric kernel depending on bandwidth parameters $h$. In the semi-supervised approach of Zhu et al. (2003), the locally constant estimate $\widehat{m}(x)$ is formed using not only the labeled data, but also using the estimates at the unlabeled points. Suppose that the first $n$ data points $(X_1, Y_1), \ldots, (X_n, Y_n)$ are labeled, and the next $u = N - n$ points are unlabeled, $X_{n+1}, \ldots, X_{n+u}$. The semi-supervised regression estimate is then $(\widehat{m}(X_1), \widehat{m}(X_2), \ldots, \widehat{m}(X_N))$ given by

$$\widehat{m} = \arg\min_m \sum_{i=1}^N \sum_{j=1}^N K_h(X_i, X_j)(m(X_i) - m(X_j))^2 \tag{15}$$

where the minimization is carried out subject to the constraint $m(X_i) = Y_i$, $i = 1, \ldots, n$. Thus, the estimates are coupled, unlike the standard kernel regression estimate (14) where the estimate at each point $x$ can be formed independently, given the labeled data.

The estimator can be written in closed form as a linear smoother $\widehat{m} = C^{-1} B Y = G Y$ where $\widehat{m} = (\widehat{m}(X_{n+1}), \ldots, m(X_{n+u}))^T$ is the vector of estimates over the unlabeled test points, and $Y = (Y_1, \ldots, Y_n)^T$ is vector of labeled values. The $(N-n) \times (N-n)$ matrix $C$ and the $(N-n) \times n$ matrix $B$ denote blocks of the combinatorial Laplacian on the data graph corresponding to the labeled and unlabeled data:

$$\Delta = \begin{pmatrix} A & B^T \\ B & C \end{pmatrix} \tag{16}$$

where the Laplacian $\Delta = \Delta_{ij}$ has entries

$$\Delta_{ij} = \begin{cases} \sum_k K_h(X_i, X_k) & \text{if } i = j \\ -K_h(X_i, X_j) & \text{otherwise.} \end{cases} \tag{17}$$

This expresses the *effective kernel G* in terms of geometric objects such as heat kernels for the discrete diffusion equations (Smola and Kondor, 2003).

This estimator assumes the noise is zero, since $\widehat{m}(X_i) = Y_i$ for $i = 1, \ldots, n$. To work in the standard model $Y = m(X) + \epsilon$, the natural extension of the harmonic function approach is *manifold regularization* (Belkin et al., 2005; Sindhwani et al., 2005; Tsang and Kwok, 2006). Here the estimator is chosen to minimize the regularized empirical risk functional

$$\mathcal{R}_\gamma(m) = \sum_{i=1}^N \sum_{j=1}^n K_H(X_i, X_j) \left(Y_j - m(X_i)\right)^2 + \gamma \sum_{i=1}^N \sum_{j=1}^N K_H(X_i, X_j) \left(m(X_j) - m(X_i)\right)^2 \tag{18}$$

where $H$ is a matrix of bandwidths and $K_H(X_i, X_j) = K(H^{-1/2}(X_i - X_j))$. When $\gamma = 0$ the standard kernel smoother is obtained. The regularization term is

$$\mathcal{J}(m) \equiv \sum_{i=1}^N \sum_{j=1}^N K_H(X_i, X_j) \left(m(X_j) - m(X_i)\right)^2 = 2m^T \Delta m \tag{19}$$

where $\Delta$ is the combinatorial Laplacian associated with $K_H$. This regularization term is motivated by the semi-supervised smoothness assumption—it favors functions $m$ for which $m(X_i)$ is close to $m(X_j)$ when $X_i$ and $X_j$ are similar, according to the kernel function. The name manifold regularization is justified by the fact that $\frac{1}{2}\mathcal{J}(m) \to \int_{\mathcal{M}} \|\nabla m(x)\|^2 \, d_{\mathcal{M}}x$, the energy of $m$ over the manifold. While this regularizer has primarily been used for SVM classifiers (Belkin et al., 2005), it can be used much more generally. For an appropriate choice of $\gamma$, minimizing the functional (18) can be expected to give essentially the same results as the harmonic function approach that minimizes (15).

**Theorem 2.** *Suppose that $D \geq 2$. Let $\widetilde{m}_{H,\gamma}$ minimize (18), and let $\Delta_{p,H}$ be the differential operator defined by*

$$\Delta_{p,H} f(x) = \frac{1}{2}\text{trace}(\mathcal{H}_f(x)H) + \frac{\nabla p(x)^T H \nabla f(x)}{p(x)}. \tag{20}$$

*Then the asymptotic MSE of $\widetilde{m}_{H,\gamma}(x)$ is*

$$\widetilde{M} = \frac{c_1 \mu \sigma^2}{n(\mu + \gamma) p(x) |H|^{1/2}} + \left(\frac{c_2(\mu + \gamma)}{\mu} \left(I - \frac{\gamma}{\mu}\Delta_{p,H}\right)^{-1} \Delta_{p,H} m(x)\right)^2 + o(\text{tr}(H)) \tag{21}$$

*where $\mu = \mathbb{P}(R_i = 1)$.*

Note that the bias of the standard kernel estimator, in the notation of this theorem, is $b(x) = c_2 \Delta_{p,H} m(x)$, and the variance is $V(x) = c_1/np(x)|H|^{1/2}$. Thus, this result agrees with the standard supervised MSE in the special case $\gamma = 0$. It follows from this theorem that $\widetilde{M} = M + o(\text{tr}(H))$ where $M$ is the usual MSE for a kernel estimator. Therefore, the minimum of $\widetilde{M}$ has the same leading order in $H$ as the minimum of $M$.

The proof is given in the full version of the paper. The implication of this theorem is that the estimator that uses Laplacian regularization has the same rate of convergence as the usual kernel estimator, and thus the unlabeled data have not improved the estimator asymptotically.

# 5 Semi-Supervised Methods With Improved Rates

The previous result is negative, in the sense that it shows unlabeled data do not help to improve the rate of convergence. This is because the bias and variance of a manifold regularized kernel estimator are of the same order in $H$ as the bias and variance of standard kernel regression. We now demonstrate how improved rates of convergence can be obtained by formulating and exploiting appropriate SSS assumptions. We describe three different approaches: semi-supervised bias reduction, improved bandwidth selection, and averaging over level sets.

## 5.1 Semi-Supervised Bias Reduction

We first show a positive result by formulating an SSS assumption that links the shape of $p$ to the shape of $m$ by positing a relationship between the Hessian $\mathcal{H}_m$ of $m$ and the Hessian $\mathcal{H}_p$ of $p$. Under this SSS assumption, we can improve the rate of convergence by reducing the bias.

To illustrate the idea, take $p(x)$ known (i.e., $N = \infty$) and suppose that $\mathcal{H}_m(x) = \mathcal{H}_p(x)$. Define

$$\widetilde{m}_n(x) = \widehat{m}_n(x) - \frac{1}{2}\mu_2^2(K)\mathrm{tr}(\mathcal{H}_m(x)H) \tag{22}$$

where $\widehat{m}_n(x)$ is the local linear estimator.

**Theorem 3.** *The risk of $\widetilde{m}_n(x)$ is $O\left(n^{-8/(8+D)}\right)$.*

*Proof.* First note that the variance of the estimator $\widetilde{m}_n$, conditional on $X_1, \ldots, X_n$, is $\mathrm{Var}(\widetilde{m}_n(x)|X_1, \ldots, X_n) = \mathrm{Var}(\widehat{m}_n(x)|X_1, \ldots, X_n)$. Now, the term $\frac{1}{2}\mu_2^2(K)\mathrm{tr}(\mathcal{H}_m(x)H)$ is precisely the bias of the local linear estimator, under the SSS assumption that $\mathcal{H}_p(x) = \mathcal{H}_m(x)$. Thus, the first order bias term has been removed. The result now follows from the fact that the next term in the bias of the local linear estimator is of order $O(\mathrm{tr}(H)^4)$. $\square$

By assuming $2\ell$ derivatives are matched, we get the rate $n^{-(4+4\ell)/(4+4\ell+D)}$. When $p$ is estimated from the data, the risk will be inflated by $N^{-4/(4+D)}$ assuming standard smoothness assumptions on $p$. This term will not dominate the improved rate $n^{-(4+4\ell)/(4+4\ell+D)}$ as long as $N > n^\ell$. The assumption that $\mathcal{H}_m = \mathcal{H}_p$ can be replaced by the more realistic assumption that $\mathcal{H}_m = g(p; \beta)$ for some parameterized family of functions $g(\cdot; \beta)$. Semiparametric methods can then be used to estimate $\beta$. This approach is taken in the following section.

## 5.2 Improved Bandwidth Selection

Let $\widehat{H}$ be the estimated bandwidth using the labeled data. We will now show how a bandwidth $\widehat{H}^*$ can be estimated using the labeled and unlabeled data together, such that, under appropriate assumptions,

$$\limsup_{n\to\infty} \frac{|R(\widehat{H}^*) - R(H^*)|}{|R(\widehat{H}) - R(H^*)|} = 0, \text{ where } H^* = \arg\min_H R(H). \tag{23}$$

Therefore, the unlabeled data allow us to construct an estimator that gets closer to the oracle risk. The improvement is weaker than the bias adjustment method. But it has the virtue that the optimal local linear rate is maintained even if the proposed model linking $\mathcal{H}_m$ to $p$ is incorrect.

We begin in one dimension to make the ideas clear. Let $\widehat{m}_H$ denote the local linear estimator with bandwidth $H \in \mathbb{R}$, $H > 0$. To use the unlabeled data, note that the optimal (global) bandwidth is $H^* = (c_2 B/(4nc_1 A))^{1/5}$ where $A = \int m''(x)^2 dx$ and $B = \int dx/p(x)$. Let $\widehat{p}(x)$ be the kernel density estimator of $p$ using $X_1, \ldots, X_N$ and bandwidth $h = O(N^{-1/5})$. We assume

(SSS) $\quad m''(x) = G_\theta(p)$ for some function $G$ depending on finitely many parameters $\theta$.

Now let $\widehat{m''(x)} = G_{\widehat{\theta}}(\widehat{p})$, and define $\widehat{H}^* = \left(\frac{c_2 \widehat{B}}{4nc_1 \widehat{A}}\right)^{1/5}$ where $\widehat{A} = \int (\widehat{m''(x)})^2 dx$ and $\widehat{B} = \int dx/\widehat{p}(x)$.

**Theorem 4.** *Suppose that $\widehat{m''(x)} - m''(x) = O_P(N^{-\beta})$ where $\beta > \frac{2}{5}$. Let $N = N(n) \to \infty$ as $n \to \infty$. If $N/n^{1/4} \to \infty$, then*

$$\limsup_{n\to\infty} \frac{|R(\widehat{H}^*) - R(H^*)|}{|R(\widehat{H}) - R(H^*)|} = 0. \tag{24}$$

*Proof.* The risk is

$$R(H) = c_1 H^4 \int (m''(x))^2 dx + \frac{c_2}{nH} \int \frac{dx}{p(x)} + o\left(\frac{1}{nH}\right). \tag{25}$$

The oracle bandwidth is $H^* = c_3/n^{1/5}$ and then $R(H^*) = O(n^{-4/5})$. Now let $\widehat{H}$ be the bandwidth estimated by cross-validation. Then, since $R'(H^*) = 0$ and $H^* = O(n^{-1/5})$, we have

$$R(\widehat{H}) = \frac{(\widehat{H} - H^*)^2}{2} R''(H^*) + O(|\widehat{H} - H^*|^3) \tag{26}$$

$$= \frac{(\widehat{H} - H^*)^2}{2} O(n^{-2/5}) + O(|\widehat{H} - H^*|^3). \tag{27}$$

From Girard (1998), $\widehat{H} - H^* = O_P(n^{-3/10})$. Hence, $R(\widehat{H}) - R(H^*) = O_P(n^{-1})$. Also, $\widehat{p}(x) - p(x) = O(N^{-2/5})$. Since $\widehat{m''(x)} - m''(x) = O_P(N^{-\beta})$,

$$\widehat{H}^* - H^* = O_P\left(\frac{N^{-2/5}}{n^{1/5}}\right) + O_P\left(\frac{N^{-\beta}}{n^{1/5}}\right). \tag{28}$$

The first term is $o_P(n^{-3/10})$ since $N > n^{1/4}$. The second term is $o_P(n^{-3/10})$ since $\beta > 2/5$. Thus $R(\widehat{H}^*) - R(H^*) = o_P(1/n)$ and the result follows. $\square$

The proof in the multidimensional case is essentially the same as in the one dimensional case, except that we use the multivariate version of Girard's result, namely, $H_* - \widehat{H} = O_P(n^{-(D+2)/(2(D+4))})$. This leads to the following result.

**Theorem 5.** *Let $N = N(n)$. If $N/n^{D/4} \to \infty$, $\widehat{\theta} - \theta = O_P(N^{-\beta})$ for some $\beta > \frac{2}{4+D}$ then*

$$\limsup_{n\to\infty} \frac{|R(\widehat{H}^*) - R(H^*)|}{|R(\widehat{H}) - R(H^*)|} = 0. \tag{29}$$

## 5.3 Averaging over Level Sets

Recall that SSS is motivated by the intuition that high density level sets should correspond to clusters of similar objects. Another approach to quantifying SSS is to make this cluster assumption explicit. Rigollet (2006) shows one way to do this in classification. Here we focus on regression.

Suppose that $L = \{x \quad p(x) > \lambda\}$ can be decomposed into a finite number of connected, compact, convex sets $C_1, \ldots, C_g$ where $\lambda$ is chosen so that $L^c$ has negligible probability. For $N$ large we can replace $L$ with $L = \{x \quad \widehat{p}(x) > \lambda\}$ with small loss in accuracy, where $\widehat{p}$ is an estimate of $p$ using the unlabeled data; see Rigollet (2006) for details. Let $k_j = \sum_{i=1}^n I(X_i \in C_j)$ and for $x \in C_j$ define

$$\widehat{m}(x) = \frac{\sum_{i=1}^n Y_i I(X_i \in C_j)}{k_j}. \tag{30}$$

Thus, $\widehat{m}(x)$ simply averages the labels of the data that fall in the set to which $x$ belongs. If the regression function is slowly varying in over this set, the risk should be small. A similar estimator is considered by Cortes and Mohri (2006), but they do not provide estimates of the risk.

**Theorem 6.** *The risk of $\widehat{m}(x)$ for $x \in L \cap C_j$ is bounded by*

$$O\left(\frac{1}{n\pi_j}\right) + O\left(\delta_j^2 \xi_j^2\right) \tag{31}$$

*where $\delta_j = \sup_{x \in C_j} \|\nabla m(x)\|$, $\xi_j = \text{diameter}(C_j)$ and $\pi_j = \mathbb{P}(X \in C_j)$.*

*Proof.* Since the $k_j$ are Binomial, $k_j = n\pi_j + o(1)$ almost surely. Thus, the variance of $\widehat{m}(x)$ is $O(1/(n\pi_j))$. The mean, given $X_1, \ldots, X_n$, is

$$\frac{1}{k_j} \sum_{i \, X_i \in C_j} m(X_i) = m(x) + \frac{1}{k_j} \sum_{i \, X_i \in C_j} (m(X_i) - m(x)). \tag{32}$$

Now $m(X_i) - m(x) = (X_j - x)^T \nabla m(u_i)$ for some $u_i$ between $x$ and $X_i$. Hence, $|m(X_i) - m(x)| \leq \|X_j - x\| \sup_{x \in C_j} \|\nabla m(x)\|$ and so the bias is bounded by $\delta_j \xi_j$. $\square$

This result reveals an interesting bias-variance tradeoff. Making $\lambda$ smaller decreases the variance and increases the bias. Suppose the two terms are balanced at $\lambda = \lambda_*$. Then we will beat the usual rate of convergence if $\pi_j(\lambda_*) > n^{-D/(4+D)}$.

# 6 Conclusion

Semi-supervised methods have been very successful in many problems. Our results suggest that the standard explanations for this success are not correct. We have indicated some new approaches to understanding and exploiting the relationship between the labeled and unlabeled data. Of course, we make no claim that these are the only ways of incorporating unlabeled data. But our results indicate that decoupling the manifold assumption and the semi-supervised smoothness assumption is crucial to clarifying the problem.

# 7 Acknowlegments

We thank Partha Niyogi for several interesting discussions. This work was supported in part by NSF grant CCF-0625879.

## Footnotes

[1]We write $a_n \asymp b_n$ to mean that $a_n/b_n$ is bounded away from 0 and infinity for large $n$. We have suppressed some technicalities such as moment assumptions on $\epsilon = Y - m(X)$.

# References

BELKIN, M., NIYOGI, P. and SINDHWANI, V. (2005). On manifold regularization. In *Proceedings of the Tenth International Workshop on Artificial Intelligence and Statistics (AISTAT 2005)*.

BICKEL, P. and LI, B. (2006). Local polynomial regression on unknown manifolds. Tech. rep., Department of Statistics, UC Berkeley.

CASTELLI, V. and COVER, T. (1996). The relative value of labeled and unlabeled samples in pattern recognition with an unknown mixing parameter. *IEEE Trans. on Info. Theory* **42** 2101–2117.

CORTES, C. and MOHRI, M. (2006). On transductive regression. In *Advances in Neural Information Processing Systems (NIPS)*, vol. 19.

FAN, J. (1993). Local linear regression smoothers and their minimax efficiencies. *The Annals of Statistics* **21** 196–216.

GIRARD, D. (1998). Asymptotic comparison of (partial) cross-validation, gcv and randomized gcv in nonparametric regression. *Ann. Statist.* **12** 315–334.

LEVINA, E. and BICKEL, P. (2005). Maximum likelihood estimation of intrinsic dimension. In *Advances in Neural Information Processing Systems (NIPS)*, vol. 17.

NIYOGI, P. (2007). Manifold regularization and semi-supervised learning: Some theoretical analyses. Tech. rep., Departments of Computer Science and Statistics, University of Chicago.

RIGOLLET, P. (2006). Generalization error bounds in semi-supervised classification under the cluster assumption. *arxiv.org/math/0604233* .

SINDHWANI, V., NIYOGI, P., BELKIN, M. and KEERTHI, S. (2005). Linear manifold regularization for large scale semi-supervised learning. In *Proc. of the 22nd ICML Workshop on Learning with Partially Classified Training Data*.

SMOLA, A. and KONDOR, R. (2003). Kernels and regularization on graphs. In *Conference on Learning Theory, COLT/KW*.

TSANG, I. and KWOK, J. (2006). Large-scale sparsified manifold regularization. In *Advances in Neural Information Processing Systems (NIPS)*, vol. 19.

ZHOU, D., BOUSQUET, O., LAL, T., WESTON, J. and SCHÖLKOPF, B. (2004). Learning with local and global consistency. In *Advances in Neural Information Processing Systems (NIPS)*, vol. 16.

ZHU, X. (2006). Semi-supervised learning literature review. Tech. rep., University of Wisconsin.

ZHU, X., GHAHRAMANI, Z. and LAFFERTY, J. (2003). Semi-supervised learning using Gaussian fields and harmonic functions. In *ICML-03, 20th International Conference on Machine Learning*.

